# A Polygonal Line Algorithm for Constructing Principal Curves

**Balázs Kégl, Adam Krzyżak**
Dept. of Computer Science
Concordia University
1450 de Maisonneuve Blvd. W.
Montreal, Canada H3G 1M8
kegl@cs.concordia.ca
krzyzak@cs.concordia.ca

**Tamás Linder**
Dept. of Mathematics
and Statistics
Queen's University
Kingston, Ontario
Canada K7L 3N6
linder@mast.queensu.ca

**Kenneth Zeger**
Dept. of Electrical and
Computer Engineering
University of California
San Diego, La Jolla
CA 92093-0407
zeger@ucsd.edu

## Abstract

Principal curves have been defined as "self consistent" smooth curves which pass through the "middle" of a $d$-dimensional probability distribution or data cloud. Recently, we [1] have offered a new approach by defining principal curves as continuous curves of a given length which minimize the expected squared distance between the curve and points of the space randomly chosen according to a given distribution. The new definition made it possible to carry out a theoretical analysis of learning principal curves from training data. In this paper we propose a practical construction based on the new definition. Simulation results demonstrate that the new algorithm compares favorably with previous methods both in terms of performance and computational complexity.

## 1 Introduction

Hastie [2] and Hastie and Stuetzle [3] (hereafter HS) generalized the self consistency property of principal components and introduced the notion of *principal curves*. Consider a $d$-dimensional random vector $\mathbf{X} = (X^{(1)}, \dots, X^{(d)})$ with finite second moments, and let $\mathbf{f}(t) = (f_1(t), \dots, f_d(t))$ be a smooth curve in $\mathcal{R}^d$ parameterized by $t \in \mathcal{R}$. For any $\mathbf{x} \in \mathcal{R}^d$ let $t_f(\mathbf{x})$ denote the parameter value $t$ for which the distance between $\mathbf{x}$ and $\mathbf{f}(t)$ is minimized. By the HS definition, $\mathbf{f}(t)$ is a principal curve if it does not intersect itself and is self consistent, that is, $\mathbf{f}(t) = E(\mathbf{X}|t_f(\mathbf{X}) = t)$. Intuitively speaking, self-consistency means that each point of $\mathbf{f}$ is the average (under the distribution of $\mathbf{X}$) of points that project there. Based on their defining property HS developed an algorithm for constructing principal curves for distributions or data sets, and described an application in the Stanford Linear Collider Project [3].

Principal curves have been applied by Banfield and Raftery [4] to identify the outlines of ice floes in satellite images. Their method of clustering about principal curves led to a fully automatic method for identifying ice floes and their outlines. On the theoretical side, Tibshirani [5] introduced a semiparametric model for principal curves and proposed a method for estimating principal curves using the EM algorithm. Recently, Delicado [6] proposed yet another definition based on a property of the first principal components of multivariate normal distributions. Close connections between principal curves and Kohonen's self-organizing maps were pointed out by Mulier and Cherkassky [7]. Self-organizing maps were also used by Der et al. [8] for constructing principal curves.

There is an unsatisfactory aspect of the definition of principal curves in the original HS paper as well as in subsequent works. Although principal curves have been defined to be *nonparametric*, their existence for a given distribution or probability density is an open question, except for very special cases such as elliptical distributions. This also makes it difficult to theoretically analyze any learning schemes for principal curves.

Recently, we [1] have proposed a new definition of principal curves which resolves this problem. In the new definition, a curve $\mathbf{f}^*$ is called a principal curve of length $L$ for $\mathbf{X}$ if $\mathbf{f}^*$ minimizes $\Delta(\mathbf{f}) = E\left[\inf_t \|\mathbf{X} - \mathbf{f}(t)\|^2\right] = E\|\mathbf{X} - \mathbf{f}(t_\mathbf{f}(\mathbf{X}))\|^2$, the expected squared distance between $\mathbf{X}$ and the curve, over all curves of length less than or equal to $L$. It was proved in [1] that for any $\mathbf{X}$ with finite second moments there always exists a principal curve in the new sense.

A theoretical algorithm has also been developed to estimate principal curves based on a common model in statistical learning theory (e.g. see [9]). Suppose that the distribution of $\mathbf{X}$ is concentrated on a closed and bounded convex set $K \subset \mathcal{R}^d$, and we are given $n$ training points $\mathbf{X}_1, \ldots, \mathbf{X}_n$ drawn independently from the distribution of $\mathbf{X}$. Let $S$ denote the family of curves taking values in $K$ and having length not greater than $L$. For $k \geq 1$ let $S_k$ be the set of polygonal (piecewise linear) curves in $K$ which have $k$ segments and whose lengths do not exceed $L$. Let

$$\Delta(\mathbf{x}, \mathbf{f}) = \min_t \|\mathbf{x} - \mathbf{f}(t)\|^2 \tag{1}$$

denote the squared distance between $\mathbf{x}$ and $\mathbf{f}$. For any $\mathbf{f} \in S$ the empirical squared error of $\mathbf{f}$ on the training data is the sample average $\Delta_n(\mathbf{f}) = \frac{1}{n}\sum_{i=1}^n \Delta(\mathbf{X}_i, \mathbf{f})$. Let the theoretical algorithm choose an $\mathbf{f}_{k,n} \in S_k$ which minimizes the empirical error, i.e, let $\mathbf{f}_{k,n} = \arg\min_{\mathbf{f}\in S_k} \Delta_n(\mathbf{f})$. It was shown in [1] that if $k$ is chosen to be proportional to $n^{1/3}$, then the expected squared loss of the empirically optimal polygonal curve with $k$ segments and length at most $L$ converges, as $n \to \infty$, to the squared loss of the principal curve of length $L$ at a rate $\Delta(\mathbf{f}_{k,n}) - \Delta(\mathbf{f}^*) = O(n^{-1/3})$.

Although amenable to theoretical analysis, the algorithm in [1] is computationally burdensome for implementation. In this paper we develop a suboptimal algorithm for learning principal curves. This practical algorithm produces polygonal curve approximations to the principal curve just as the theoretical method does, but global optimization is replaced by a less complex iterative descent method. We give simulation results and compare our algorithm with previous work. In general, on examples considered by HS the performance of the new algorithm is comparable with the HS algorithm, while it proves to be more robust to changes in the data generating model.

## 2  A Polygonal Line Algorithm

Given a set of data points $\mathcal{X}_n = \{\mathbf{x}_1, \ldots, \mathbf{x}_n\} \subset \mathcal{R}^d$, the task of finding the polygonal curve with $k$ segments and length $L$ which minimizes $\frac{1}{n}\sum_{i=1}^n \Delta(\mathbf{x}_i, \mathbf{f})$ is computationally difficult. We propose a suboptimal method with reasonable complexity. The basic idea is to start with a straight line segment $\mathbf{f}_{1,n}$ ($k = 1$) and in each iteration of the algorithm to increase

the number of segments by one by adding a new vertex to the polygonal curve $\mathbf{f}_{k,n}$ produced by the previous iteration. After adding a new vertex, the positions of all vertices are updated in an inner loop.

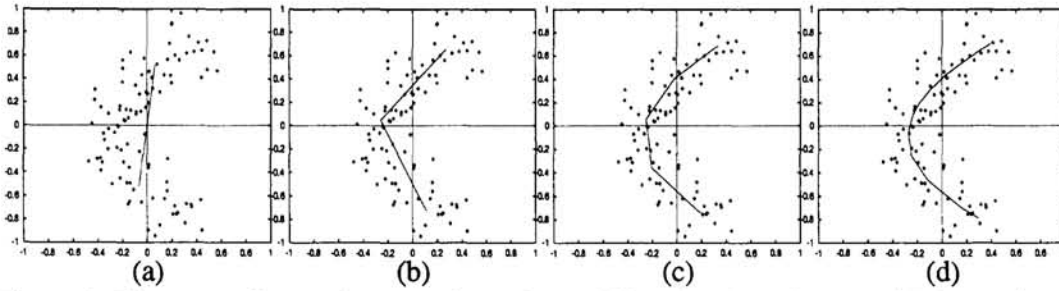

Figure 1: The curves $\mathbf{f}_{k,n}$ produced by the polygonal line algorithm for $n = 100$ data points. The data was generated by adding independent Gaussian errors to both coordinates of a point chosen randomly on a half circle. (a) $\mathbf{f}_{1,n}$, (b) $\mathbf{f}_{2,n}$, (c) $\mathbf{f}_{4,n}$, (d) $\mathbf{f}_{11,n}$ (the output of the algorithm).

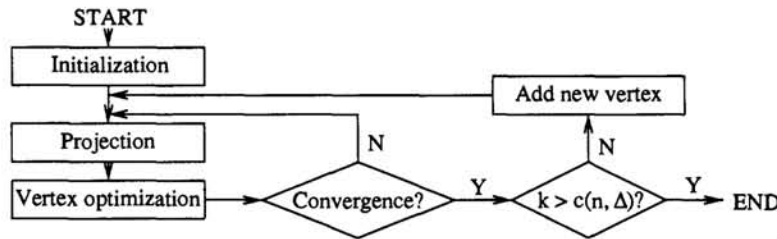

Figure 2: The flow chart of the polygonal line algorithm.

The inner loop consists of a projection step and an optimization step. In the projection step the data points are partitioned into "Voronoi regions" according to which segment or vertex they project. In the optimization step the new position of each vertex is determined by minimizing an average squared distance criterion penalized by a measure of the local curvature. These two steps are iterated until convergence is achieved and $f_{k,n}$ is produced. Then a new vertex is added.

The algorithm stops when $k$ exceeds a threshold $c(n, \Delta)$. This stopping criterion is based on a heuristic complexity measure, determined by the number segments $k$, the number of data points $n$, and the average squared distance $\Delta_n(\mathbf{f}_{k,n})$.

THE INITIALIZATION STEP. To obtain $\mathbf{f}_{1,n}$, take the shortest segment of the first principal component line which contains all of the projected data points.

THE PROJECTION STEP. Let $\mathbf{f}$ denote a polygonal curve with vertices $\mathbf{v}_1, \ldots, \mathbf{v}_{k+1}$ and closed line segments $\mathbf{s}_1, \ldots, \mathbf{s}_k$, such that $\mathbf{s}_i$ connects vertices $\mathbf{v}_i$ and $\mathbf{v}_{i+1}$. In this step the data set $X_n$ is partitioned into (at most) $2k + 1$ disjoint sets $V_1, \ldots, V_{k+1}$ and $S_1, \ldots, S_k$, the Voronoi regions of the vertices and segments of $\mathbf{f}$, in the following manner. For any $\mathbf{x} \in \mathcal{R}^d$ let $\Delta(\mathbf{x}, \mathbf{s}_i)$ be the squared distance from $\mathbf{x}$ to $\mathbf{s}_i$ (see definition (1)), and let $\Delta(\mathbf{x}, \mathbf{v}_i) = \|\mathbf{x} - \mathbf{v}_i\|^2$. Then let

$$V_i = \{\mathbf{x} \in X_n : \Delta(\mathbf{x}, \mathbf{v}_i) = \Delta(\mathbf{x}, \mathbf{f}), \quad \Delta(\mathbf{x}, \mathbf{v}_i) < \Delta(\mathbf{x}, \mathbf{v}_m), m = 1, \ldots, i-1\}.$$

Upon setting $V = \bigcup_{i=1}^{k+1} V_i$, the $S_i$ sets are defined by

$$S_i = \{\mathbf{x} \in X_n : \mathbf{x} \notin V, \Delta(\mathbf{x}, \mathbf{s}_i) = \Delta(\mathbf{x}, \mathbf{f}), \quad \Delta(\mathbf{x}, \mathbf{s}_i) < \Delta(\mathbf{x}, \mathbf{s}_m), m = 1, \ldots, i-1\}.$$

The resulting partition is illustrated in Figure 3.

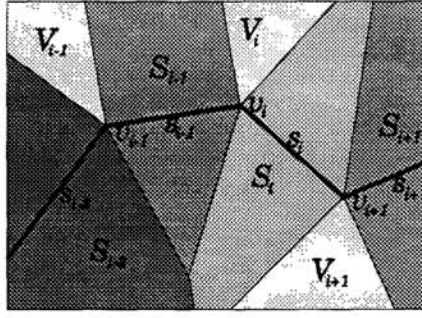

Figure 3: The Voronoi partition induced by the vertices and segments of **f**

THE VERTEX OPTIMIZATION STEP. In this step we iterate over the vertices, and relocate each vertex while all the others are kept fixed. For each vertex, we minimize $\Delta_n(\mathbf{v}_i) + \lambda_p P(\mathbf{v}_i)$, a local average squared distance criterion penalized by a measure of the local curvature by using a gradient (steepest descent) method.

The local measure of the average squared distance is calculated from the data points which project to $\mathbf{v}_i$ or to the line segment(s) starting at $\mathbf{v}_i$ (see Projection Step). Accordingly, let $\sigma_+(\mathbf{v}_i) = \sum_{\mathbf{x} \in S_i} \Delta(\mathbf{x}, \mathbf{s}_i)$, $\sigma_-(\mathbf{v}_i) = \sum_{\mathbf{x} \in S_{i-1}} \Delta(\mathbf{x}, \mathbf{s}_{i-1})$, and $\nu(\mathbf{v}_i) = \sum_{\mathbf{x} \in V_i} \Delta(\mathbf{x}, \mathbf{v}_i)$. Now define the local average squared distance as a function of $\mathbf{v}_i$ by

$$\Delta_n(\mathbf{v}_i) = \begin{cases} \dfrac{\nu(\mathbf{v}_i) + \sigma_+(\mathbf{v}_i)}{|V_i| + |S_i|} & \text{if } i = 1 \\[2ex] \dfrac{\sigma_-(\mathbf{v}_i) + \nu(\mathbf{v}_i) + \sigma_+(\mathbf{v}_i)}{|S_{i-1}| + |V_i| + |S_i|} & \text{if } 1 < i < k+1 \\[2ex] \dfrac{\sigma_-(\mathbf{v}_i) + \nu(\mathbf{v}_i)}{|S_{i-1}| + |V_i|} & \text{if } i = k+1. \end{cases} \qquad (2)$$

In the theoretical algorithm the average squared distance $\Delta_n(\mathbf{x}, \mathbf{f})$ is minimized subject to the constraint that **f** is a polygonal curve with $k$ segments and length not exceeding $L$. One could use a Lagrangian formulation and attempt to find a new position for $\mathbf{v}_i$ (while all other vertices are fixed) such that the penalized squared error $\Delta_n(\mathbf{f}) + \lambda l(\mathbf{f})^2$ is minimum. However, we have observed that this approach is very sensitive to the choice of $\lambda$, and reproduces the estimation bias of the HS algorithm which flattens the curve at areas of high curvature. So, instead of directly penalizing the lengths of the line segments, we chose to penalize sharp angles to obtain a smooth curve solution. Nonetheless, note that if only one vertex is moved at a time, penalizing sharp angles will indirectly penalize long line segments. At inner vertices $\mathbf{v}_i$, $3 \le i \le k-1$ we penalize the sum of the cosines of the three angles at vertices $\mathbf{v}_{i-1}$, $\mathbf{v}_i$, and $\mathbf{v}_{i+1}$. The cosine function was picked because of its regular behavior around $\pi$, which makes it especially suitable for the steepest descent algorithm. To make the algorithm invariant under scaling, we multiply the cosines by the squared radius of the data, that is, $r = 1/2 \max_{\mathbf{x} \in X_n, \mathbf{y} \in X_n} \|\mathbf{x} - \mathbf{y}\|$. At the endpoints and at their immediate neighbors ($\mathbf{v}_i$, $i = 1, 2, k, k+1$), where penalizing sharp angles does not translate to penalizing long line segments, the penalty on a nonexistent angle is replaced by a direct penalty on the squared length of the first (or last) segment. Formally, let $\gamma_i$ denote the angle at vertex $\mathbf{v}_i$, let $\pi(\mathbf{v}_i) = r^2(1 + \cos\gamma_i)$, let $\mu_+(\mathbf{v}_i) = \|\mathbf{v}_i - \mathbf{v}_{i+1}\|^2$, and let $\mu_-(\mathbf{v}_i) = \|\mathbf{v}_i - \mathbf{v}_{i-1}\|^2$. Then the penalty at vertex $\mathbf{v}_i$ is

$$P(\mathbf{v}_i) = \begin{cases} 2\mu_+(\mathbf{v}_i) + \pi(\mathbf{v}_{i+1}) & \text{if } i = 1 \\ \mu_-(\mathbf{v}_i) + \pi(\mathbf{v}_i) + \pi(\mathbf{v}_{i+1}) & \text{if } i = 2 \\ \pi(\mathbf{v}_{i-1}) + \pi(\mathbf{v}_i) + \pi(\mathbf{v}_{i+1}) & \text{if } 2 \le i \le k-1 \\ \pi(\mathbf{v}_{i-1}) + \pi(\mathbf{v}_i) + \mu_+(\mathbf{v}_i) & \text{if } i = k \\ \pi(\mathbf{v}_{i-1}) + 2\mu_-(\mathbf{v}_i) & \text{if } i = k+1. \end{cases}$$

One important issue is the amount of smoothing required for a given data set. In the HS algorithm one needs to set the penalty coefficient of the spline smoother, or the span of the scatterplot smoother. In our algorithm, the corresponding parameter is the curvature penalty factor $\lambda_p$. If some a priori knowledge about the distribution is available, one can use it to determine the smoothing parameter. However in the absence of such knowledge, the coefficient should be data-dependent. Intuitively, $\lambda_p$ should increase with the number of segments and the size of the average squared error, and it should decrease with the data size. Based on heuristic considerations and after carrying out practical experiments, we set $\lambda_p = \lambda'_p n^{-1/3} \Delta_n(\mathbf{f}_{k,n})^{1/2} r^{-1}$, where $\lambda'_p$ is a parameter of the algorithm, and can be kept fixed for substantially different data sets.

ADDING A NEW VERTEX. We start with the optimized $\mathbf{f}_{k,n}$ and choose the segment that has the largest number of data points projecting to it. If more then one such segment exists, we choose the longest one. The midpoint of this segment is selected as the new vertex. Formally, let $I = \{i : |S_i| \geq |S_j|, j = 1, \ldots, k\}$, and $\ell = \arg\max_{i \in I} \|\mathbf{v}_i - \mathbf{v}_{i+1}\|$. Then the new vertex is $\mathbf{v}_{new} = (\mathbf{v}_\ell + \mathbf{v}_{\ell+1})/2$.

STOPPING CONDITION. According to the theoretical results of [1], the number of segments $k$ should be proportional to $n^{1/3}$ to achieve the $O(n^{1/3})$ convergence rate for the expected squared distance. Although the theoretical bounds are not tight enough to determine the optimal number of segments for a given data size, we found that $k \sim n^{1/3}$ also works in practice. To achieve robustness we need to make $k$ sensitive to the average squared distance. The stopping condition blends these two considerations. The algorithm stops when $k$ exceeds $c(n, \Delta_n(\mathbf{f}_{k,n})) = \lambda_k n^{1/3} \Delta_n(\mathbf{f}_{k,n})^{-1/2} r$.

COMPUTATIONAL COMPLEXITY. The complexity of the inner loop is dominated by the complexity of the projection step, which is $O(nk)$. Increasing the number of segments by one at a time (as described in Section 2), and using the stopping condition of Section 2, the computational complexity of the algorithm becomes $O(n^{5/3})$. This is slightly better than the $O(n^2)$ complexity of the HS algorithm. The complexity can be dramatically decreased if, instead of adding only one vertex, a new vertex is placed at the midpoint of every segment, giving $O(n^{4/3} \log n)$, or if $k$ is set to be a constant, giving $O(n)$. These simplifications work well in certain situations, but the original algorithm is more robust.

## 3   Experimental Results

We have extensively tested our algorithm on two-dimensional data sets. In most experiments the data was generated by a commonly used (see, e.g., [3] [5] [7]) additive model $\mathbf{X} = \mathbf{Y} + \mathbf{e}$, where $\mathbf{Y}$ is uniformly distributed on a smooth planar curve (hereafter called the *generating curve*) and $\mathbf{e}$ is bivariate additive noise which is independent of $\mathbf{Y}$.

Since the "true" principal curve is not known (note that the generating curve in the model $\mathbf{X} = \mathbf{Y} + \mathbf{e}$ is in general not a principal curve either in the HS sense or in our definition), it is hard to give an objective measure of performance. For this reason, in what follows, the performance is judged subjectively, mainly on the basis of how closely the resulting curve follows the shape of the generating curve.

In general, in simulation examples considered by HS the performance of the new algorithm is comparable with the HS algorithm. Due to the data-dependence of the curvature penalty factor and the stopping condition, our algorithm turns out to be more robust to alterations in the data generating model, as well as to changes in the parameters of the particular model.

We use varying generating shapes, noise parameters, and data sizes to demonstrate the robustness of the polygonal line algorithm. All of the plots in Figure 4 show the generating curve (Generator Curve), the curve produced by our polygonal line algorithm (Principal

Curve), and the curve produced by the HS algorithm with spline smoothing (HS Principal Curve), which we have found to perform better than the HS algorithm using scatterplot smoothing. For closed generating curves we also include the curve produced by the Banfield and Raftery (BR) algorithm [4], which extends the HS algorithm to closed curves (BR Principal Curve). The two coefficients of the polygonal line algorithm are set in all experiments to the constant values $\lambda_k = 0.3$ and $\lambda'_p = 0.1$. All plots have been normalized to fit in a $2 \times 2$ square. The parameters given below refer to values before this normalization.

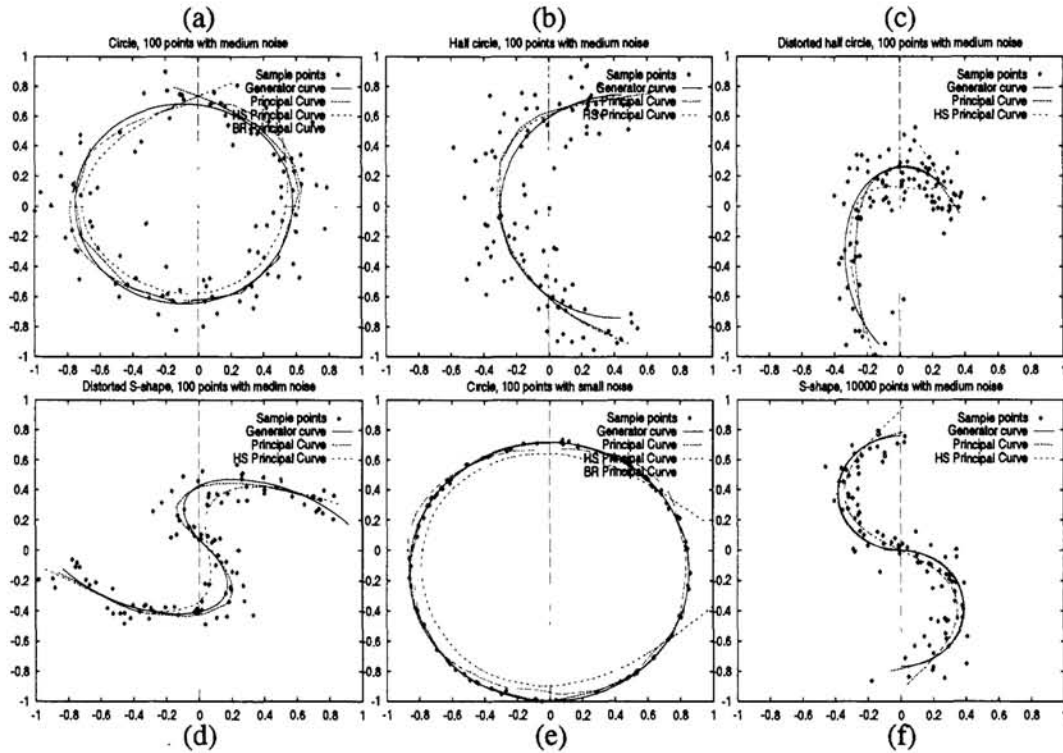

Figure 4: (a) **The Circle Example**: the BR and the polygonal line algorithm show less bias than the HS algorithm. (b) **The Half Circle Example**: the HS and the polygonal line algorithms produce similar curves. (c) and (d) **Transformed Data Sets**: the polygonal line algorithm still follows fairly closely the "distorted" shapes. (e) **Small Noise Variance** and (f) **Large Sample Size**: the curves produced by the polygonal line algorithm are nearly indistinguishable from the generating curves.

In Figure 4(a) the generating curve is a circle of radius $r = 1$, and $\mathbf{e} = (e_1, e_2)$ is a zero mean bivariate uncorrelated Gaussian with variance $E(e_i^2) = 0.04$, $i = 1, 2$. The performance of the three algorithms (HS, BR, and the polygonal line algorithm) is comparable, although the HS algorithm exhibits more bias than the other two. Note that the BR algorithm [4] has been tailored to fit closed curves and to reduce the estimation bias. In Figure 4(b), only half of the circle is used as a generating curve and the other parameters remain the same. Here, too, both the HS and our algorithm behave similarly.

When we depart from these usual settings the polygonal line algorithm exhibits better behavior than the HS algorithm. In Figure 4(c) the data set of Figure 4(b) was linearly transformed using the matrix $\left( \begin{smallmatrix} 0.6 & 0.6 \\ -1.0 & 1.2 \end{smallmatrix} \right)$. In Figure 4(d) the transformation $\left( \begin{smallmatrix} -1.0 & -1.2 \\ 1.0 & -0.2 \end{smallmatrix} \right)$ was used. The original data set was generated by an S-shaped generating curve, consisting of two half circles of unit radii, to which the same Gaussian noise was added as in Figure 4(b). In both cases the polygonal line algorithm produces curves that fit the generator curve more closely. This is especially noticeable in Figure 4(c) where the HS principal curve fails to follow the shape of the distorted half circle.

There are two situations when we expect our algorithm to perform particularly well. If the distribution is concentrated on a curve, then according to both the HS and our definitions the principal curve is the generating curve itself. Thus, if the noise variance is small, we expect both algorithms to very closely approximate the generating curve. The data in Figure 4(e) was generated using the same additive Gaussian model as in Figure 4(a), but the noise variance was reduced to $E(e_i^2) = 0.001$ for $i = 1, 2$. In this case we found that the polygonal line algorithm outperformed both the HS and the BR algorithms.

The second case is when the sample size is large. Although the generating curve is not necessarily the principal curve of the distribution, it is natural to expect the algorithm to well approximate the generating curve as the sample size grows. Such a case is shown in Figure 4(f), where $n = 10000$ data points were generated (but only a small subset of these was actually plotted). Here the polygonal line algorithm approximates the generating curve with much better accuracy than the HS algorithm.

The Java implementation of the algorithm is available at the WWW site
```
http://www.cs.concordia.ca/~grad/kegl/pcurvedemo.html
```

## 4 Conclusion

We offered a new definition of principal curves and presented a practical algorithm for constructing principal curves for data sets. One significant difference between our method and previous principal curve algorithms ([3],[4], and [8]) is that, motivated by the new definition, our algorithm minimizes a distance criterion (2) between the data points and the *polygonal curve* rather than minimizing a distance criterion between the data points and the vertices of the polygonal curve. This and the introduction of the data-dependent smoothing factor $\lambda_p$ made our algorithm more robust to variations in the data distribution, while we could keep computational complexity low.

## Acknowledgments

This work was supported in part by NSERC grant OGP000270, Canadian National Networks of Centers of Excellence grant 293, and the National Science Foundation.

## References

[1] B. Kégl, A. Krzyżak, T. Linder, and K. Zeger, "Principal curves: Learning and convergence," in *Proceedings of IEEE Int. Symp. on Information Theory*, p. 387, 1998.

[2] T. Hastie, *Principal curves and surfaces*. PhD thesis, Stanford University, 1984.

[3] T. Hastie and W. Stuetzle, "Principal curves," *Journal of the American Statistical Association*, vol. 84, no. 406, pp. 502–516, 1989.

[4] J. D. Banfield and A. E. Raftery, "Ice floe identification in satellite images using mathematical morphology and clustering about principal curves," *Journal of the American Statistical Association*, vol. 87, no. 417, pp. 7–16, 1992.

[5] R. Tibshirani, "Principal curves revisited," *Statistics and Computation*, vol. 2, pp. 183–190, 1992.

[6] P. Delicado, "Principal curves and principal oriented points," Tech. Rep. 309, Department d'Economia i Empresa, Universitat Pompeu Fabra, 1998.
```
http://www.econ.upf.es/deehome/what/wpapers/postscripts/309.pdf.
```

[7] F. Mulier and V. Cherkassky, "Self-organization as an iterative kernel smoothing process," *Neural Computation*, vol. 7, pp. 1165–1177, 1995.

[8] R. Der, U. Steinmetz, and G. Balzuweit, "Nonlinear principal component analysis," tech. rep., Institut für Informatik, Universität Leipzig, 1998.
```
http://www.informatik.uni-leipzig.de/~der/Veroeff/npcafin.ps.
```

[9] V. N. Vapnik, *The Nature of Statistical Learning Theory*. New York: Springer-Verlag, 1995.